# Recognition-based Segmentation of On-line Hand-printed Words

M. Schenkel*, H. Weissman, I. Guyon, C. Nohl, D. Henderson
AT&T Bell Laboratories, Holmdel, NJ 07733
* Swiss Federal Institute of Technology, CH-8092 Zürich

## Abstract

This paper reports on the performance of two methods for recognition-based segmentation of strings of on-line hand-printed capital Latin characters. The input strings consist of a time-ordered sequence of X-Y coordinates, punctuated by pen-lifts. The methods were designed to work in "run-on mode" where there is no constraint on the spacing between characters. While both methods use a neural network recognition engine and a graph-algorithmic post-processor, their approaches to segmentation are quite different. The first method, which we call $INSEG$ (for input segmentation), uses a combination of heuristics to identify particular pen-lifts as tentative segmentation points. The second method, which we call $OUTSEG$ (for output segmentation), relies on the empirically trained recognition engine for both recognizing characters and identifying relevant segmentation points.

## 1 INTRODUCTION

We address the problem of writer independent recognition of hand-printed words from an 80,000-word English dictionary. Several levels of difficulty in the recognition of hand-printed words are illustrated in figure 1. The examples were extracted from our databases (table 1). Except in the cases of boxed or clearly spaced characters, segmenting characters independently of the recognition process yields poor recognition performance. This has motivated us to explore recognition-based segmentation techniques.

Table 1: **Databases used for training and testing.** *DB2* contains words one to five letters long, but only four and five letter words are constrained to be legal English words. *DB3* contains legal English words of any length from an 80,000 word dictionary.

| uppercase database | data nature | pad used | training set size | test set size | approx. # of donors |
|---|---|---|---|---|---|
| *DB1* | boxed letters | AT&T | 9000 | 1500 | 250 |
| *DB2* | short words | Grid | 8000 | 1000 | 400 |
| *DB3* | English words | Wacom | - | 600 | 25 |

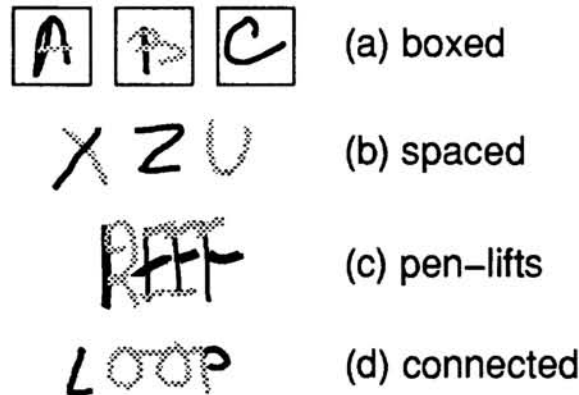

(a) boxed

(b) spaced

(c) pen–lifts

(d) connected

Figure 1: **Examples of styles that can be found in our databases:** (a) *DB*1; (b) *DB*2; (c), (d) *DB*2 and *DB*3. The line thickness or darkness is alternated at each pen-lift.

The basic principle of recognition-based segmentation is to present to the recognizer many "tentative characters". The recognition scores ultimately determine the string segmentation. We have investigated two different recognition-based segmentation methods which differ in their definition of the tentative characters, but have very similar recognition engines.

The data collection device provides pen trajectory information as a sequence of $(x, y)$ coordinates at regular time intervals (10-15 ms). We use a preprocessing technique which preserves this information by keeping a finely sampled sequence of feature vectors along the pen trajectory (Guyon et al. 1991, Weissman et al. 1992). The recognizer is a Time Delay Neural Network ($TDNN$) (Lang and Hinton 1988, Waibel et al. 1989, Guyon et al. 1991). There is one output per class, in this case 26 outputs, providing a score for all the capital letters of the Latin alphabet.

The critical step in the segmentation process is the postprocessing which disentangles various word hypotheses using the character recognition scores provided by the $TDNN$. For this purpose, we use conventional dynamic programming algorithms. In addition we use a dictionary that checks the solution and returns a list of similar legal words. The best word hypotheses, subject to this list, is again chosen by dynamic programming algorithms.

Recognition-based segmentation relies on the recognizer to give low confidence

scores for wrong tentative characters corresponding to a segmentation mistake. Recognizers trained only on valid characters usually perform poorly on such a task.

We use "segmentation-driven training" techniques which allow the training of wrong tentative characters, produced by the segmentation engine itself, as negative examples. This additional training has reduced our error rates by more than a factor of two.

In section 2 we describe the $INSEG$ method which uses tentative characters delineated by heuristic segmentation points. It is expected to be most appropriate for hand-printed capital letters since nearly all writers separate these letters by pen-lifts. This method was inspired by a similar technique used for Optical Character Recognition (OCR) (Burges et al. 1992). In section 3 we present an alternative method, $OUTSEG$, which expects the recognition engine to learn empirically (learning by examples) both to recognize characters and to identify relevant segmentation points. This second method bears similarities with the OCR methods proposed by Matan et al. (1991) or Keeler et al. (1991). In section 4 we compare the two methods and present experimental results.

## 2   SEGMENTATION IN INPUT SPACE

Figure 2 shows the different steps of the $INSEG$ process. Module 1 is used to define "tentative characters" delineated by "tentative cuts" (spaces or pen-lifts). The tentative characters are then handed to module 2 which performs the preprocessing and the scoring of the characters with a $TDNN$. The recognition results are then gathered into an interpretation graph. In module 3 the best path through that graph is found with the Viterbi algorithm.

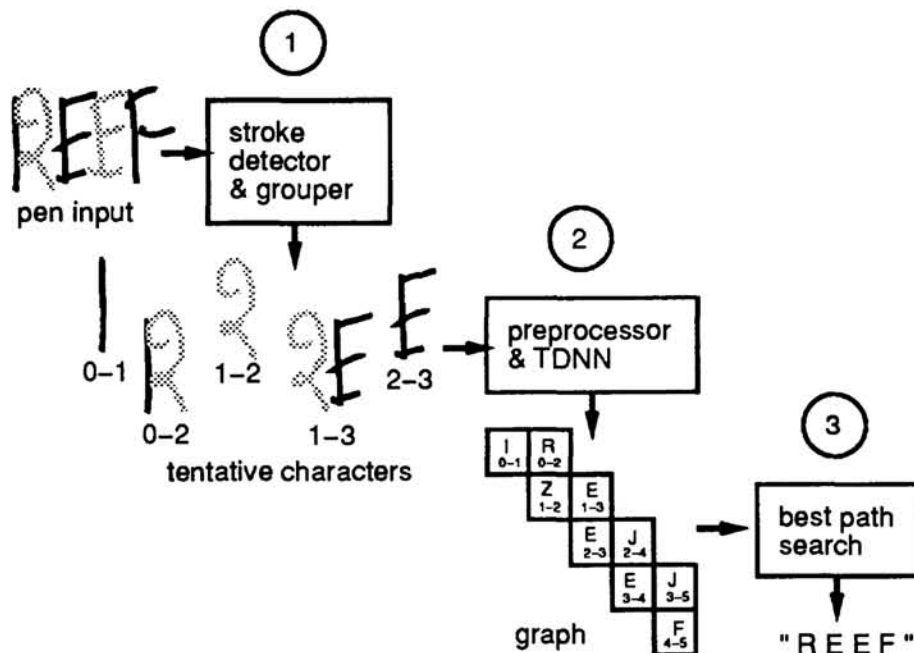

Figure 2: **Processing steps of the $INSEG$ method.**

In figure 3 we show a simplified representation of an interpretation graph built by our system. Each tentative character (denoted $\{i, j\}$) has a double index: the tentative cut $i$ at the character starting point and the tentative cut $j$ at the character end point. We denote by $X\{i, j\}$ the node associated to the score of letter $X$ for the tentative character $\{i, j\}$. A path through the graph starts at a node $X\{0, .\}$ and ends at a node $Y\{., m\}$, where 0 is the word starting point and $m$ the last pen-lift. In between, only transitions of the kind $X\{., i\} \rightarrow Y\{i, .\}$ are allowed to prevent character overlapping.

To avoid searching through too complex a graph, we need to perform some pruning. The spatial relationship between strokes is used to discard unlikely tentative cuts. For instance, strokes with a large horizontal overlap are bundled. The remaining tentative characters are then grouped in different ways to form alternative tentative characters. Tentative characters separated by a large horizontal spatial interval are never considered for grouping.

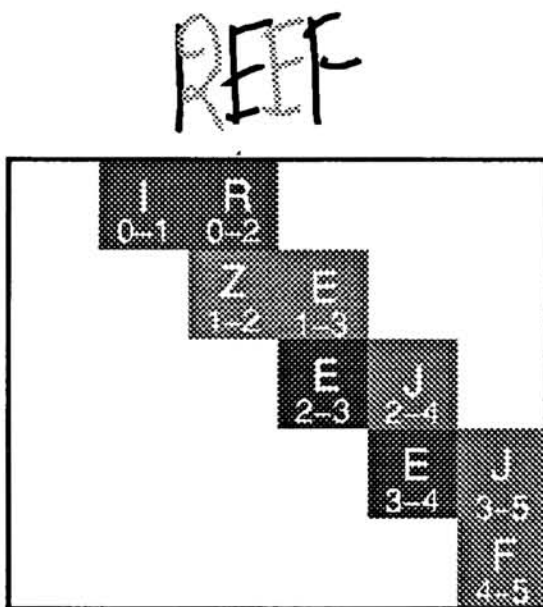

Figure 3: **Graph obtained with the input segmentation method.**
The grey shading in each box indicates the recognition scores (the darker, the stronger the recognition score and the higher the recognition confidence).

In table 2 we present the results obtained with the $TDNN$ recognizer used by Guyon et al. (1991), with 4 convolutional layers and 6,252 weights. Characters are preprocessed individually, which provides the network with a fixed dimension input.

## 3   SEGMENTATION IN OUTPUT SPACE

In contrast with $INSEG$, the $OUTSEG$ method does not rely on human designed segmentation hints: the neural network learns both recognition and segmentation features from examples.

Tentative characters are produced simply in that a window is swept over the input sequence in small steps. At each step the content of the window is taken to be a tentative character. Successive characters usually overlap considerably.

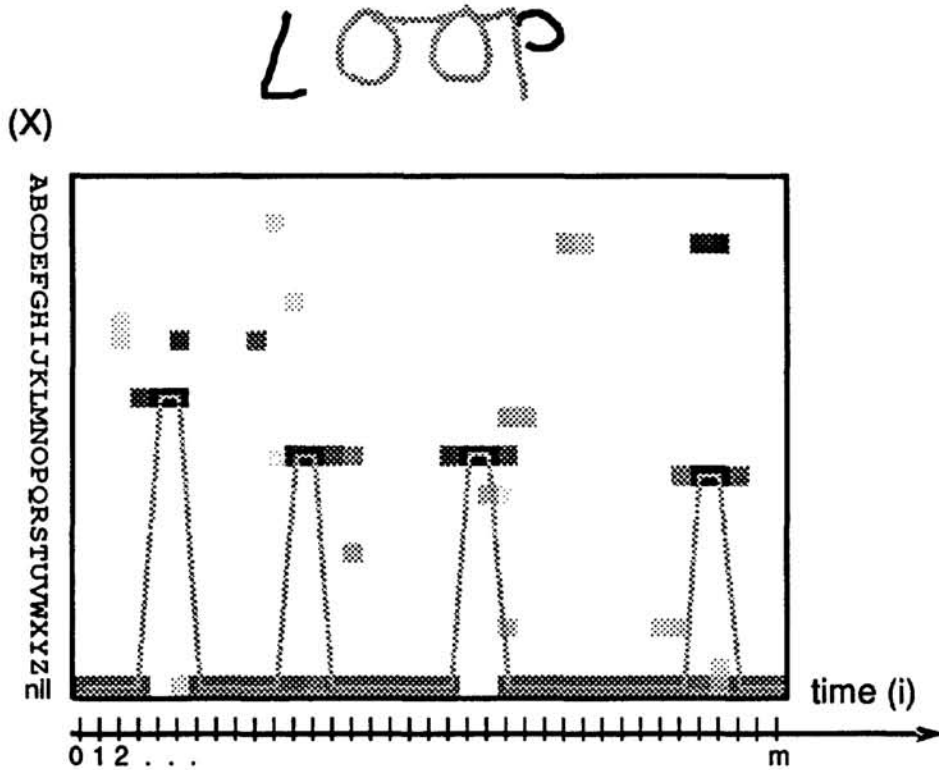

Figure 4: *TDNN* **outputs of the** *OUTSEG* **system.**
The grey curve indicates the best path through the graph, using duration modeling. The word "LOOP" was correctly recognized in spite of the ligatures which prevent segmentation on the basis of pen-lifts.

In figure 4, we show the outputs of our $TDNN$ recognizer when the word "LOOP" is processed. The main matrix is a simplified representation of our interpretation graph. Tentative character numbers $i$ ($i \in \{1, 2, ..., m\}$), run along the time direction. Each column contains the scores of all possible interpretations $X$ ($X \in \{A, B, C, ..., Z, nil\}$) of a given tentative character. The bottom line is the $nil$ interpretation score which approximates the probability that the present input is not a character (meaningless character): $P(nil\{i\}|input) = 1 - (P(A\{i\}|input) + P(B\{i\}|input) + ... + P(Z\{i\}|input))$

The connections between nodes reflect a model of character durations. A simple way of enforcing duration is to allow only the following transitions:

$$X\{i\} \quad \rightarrow \quad X\{i+1\},$$
$$nil\{i\} \quad \rightarrow \quad nil\{i+1\},$$
$$X\{i\} \quad \rightarrow \quad nil\{i+1\},$$
$$nil\{i\} \quad \rightarrow \quad X\{i+1\},$$

where $X$ stands for a certain letter. A character interpretation can be followed by

the same interpretation but cannot be followed immediately by another character interpretation: they must be separated by *nil*. This permits distinguishing between letter duration and letter repetition (such as the double "O" in our example). The best path in the graph is found by the Viterbi algorithm.

In fact, this simple pattern of connections corresponds to a Markov model of duration, with exponential decay. We implemented a slightly fancier model which allows the generation of any duration distribution (Weissman et al. 1992) to help prevent character omission or insertion. In our experiments, we selected two Poisson distributions to model character and the *nil*-class duration respectively.

We use a $TDNN$ recognizer with 3 layers and 10,817 weights. The sequence of recognition scores is obtained by sweeping the neural network over the input. Because of the convolutional structure of the $TDNN$, there are many identical computations between two successive calls of the recognizer and only about one sixth of the network connections have to be reevaluated for each new tentative character. As a consequence, although the $OUTSEG$ system processes many more tentative characters than the $INSEG$ system does, the overall computation time is about the same.

## 4    COMPARISON OF RESULTS AND CONCLUSIONS

Table 2: **Comparison of the performance of the two segmentation methods using a $TDNN$ recognizer.**

|  | Error without dictionary | | Error with dictionary | |
|---|---|---|---|---|
| on $DB2$ | % char. | % word | % char. | % word |
| $INSEG$ | 9 | 18 | 8.5 | 15 |
| $OUTSEG$ | 10 | 21 | 8 | 17 |
| on $DB3$ | % char. | % word | % char. | % word |
| $INSEG$ | 8 | 33 | 5 | 13 |
| $OUTSEG$ | 11 | 48 | 7 | 21 |

We summarize in table 2 the results obtained with our two segmentation methods. To complement the results obtained with database $DB2$, we used (without retraining) database $DB3$ as a control, containing words of any length from the English dictionary. In our current versions, $INSEG$ performs better than $OUTSEG$. The $OUTSEG$ method can handle connected letters (such as in the example of the word "LOOP" in figure 4), while the $INSEG$ method, which relies on pen lifts, cannot. But, we discovered that very few people did not separate their characters by pen lifts in the data we collected. On the other hand, an advantage of the $INSEG$ method is that it can easily be used with recognizers other than the $TDNN$, whereas the $OUTSEG$ method relies heavily on the convolutional structure of the $TDNN$ for computational efficiency.

For comparison, we substituted two other neural network recognizers to the $TDNN$. These networks use alternative input representations. The $OCR-net$ was designed for Optical Character Recognition (Le Cun et al. 1989) and uses pixel map inputs.

Its first layer performs local line orientation detection. The *orientation − net* has an architecture similar to that of the $OCR − net$, but its first layer is removed and local line orientation information, directly extracted from the pen trajectory, is transmitted to the second layer (Weissbuch and Le Cun 1992). Without a dictionary, the $OCR − net$ has an error rate more than twice that of the $TDNN$ but the *orientation − net* performs similarly. With dictionary the *orientation − net* has a 25% lower error rate than the $TDNN$. This improvement is attributed to better second and third best recognition choices, which facilitates dictionary use.

Our best results to date (tables 3) were obtained with the $INSEG$ method, using two recognizers combined with a voting scheme: the $TDNN$ and the *orientation − net*. For comparison purposes we mention the results obtained by a commercial recognizer on the same data. One should notice that our dictionary is the same as the one from which the data was drawn and is probably a larger dictionary than the one used by the commercial system. Our results are substantially better than those of the commercial system. On an absolute scale they are quite satisfactory if we take into account that the test data was not cleaned at all and that more than 20% of the errors have been identified to be patterns written in cursive, misspelled or totally illegible.

We expect the $OUTSEG$ method to work best for cursive handwriting, which does not exhibit trivial segmentation hints, but we do not have any direct evidence to support this expectation as yet. Rumelhart (1992) had success with a version of $OUTSEG$. Work is in progress to extend the capabilities of our systems to cursive writing.

Table 3: **Performance of our best system.** For comparison, we mention in parenthesis the performances obtained by a commercial recognizer on the same data. The performance of the commercial system with dictionary (marked with a *) are penalized because $DB2$ and $DB3$ include words not contained in its dictionary.

| Method | Error without dictionary | | Error with dictionary | |
|---|---|---|---|---|
| | % char. | % word | % char. | % word |
| $DB2$ | 7   (18) | 13   (29) | 7   (17*) | 10   (32*) |
| $DB3$ | 6   (20) | 23   (61) | 5   (18*) | 11   (49*) |

**Acknowledgments**

We wish to thank the entire Neural Network group at Bell Labs Holmdel for their supportive discussions. Helpful suggestions with the editing of this paper by L. Jackel and B. Boser are gratefully acknowledged. We are grateful to Anne Weissbuch, Yann Le Cun and Jan Ben for giving us their Neural Networks to try on our $INSEG$ method. We are indebted to Howard Page for providing comparison figures with the commercial recognizer. The experiments were performed with the neural network simulators of B. Boser, Y. Le Cun and L. Bottou who we thank for their help and advice.

## References

I. Guyon, P. Albrecht, Y. Le Cun, J. Denker and W. Hubbard. Design of a neural network character recognizer for a touch terminal. *Pattern Recognition*, 24(2), 1991.

H. Weissman, M. Schenkel, I. Guyon, C. Nohl and D. Henderson. Recognition-based Segmentation of On-line Run-on Handprinted Words: Input vs. Output Segmentation. Submitted to Pattern Recognition, October 1992.

K. J. Lang and G. E. Hinton. A time delay neural network architecture for speech recognition. Technical Report CMU-cs-88-152, Carnegie-Mellon University, Pittsburgh PA, 1988.

A. Waibel, T. Hanazawa, G. Hinton, K. Shikano and K. Lang. Phoneme recognition using time-delay neural networks. *IEEE Transactions on Acoustics, Speech and Signal Processing*, 37:328–339, March 1989.

C. J. C. Burges, O. Matan, Y. Le Cun, D. Denker, L. D. Jackel, C. E. Stenard, C. R. Nohl and J. I. Ben. Shortest path segmentation: A method for training neural networks to recognize character strings. In *IJCNN'92*, volume 3, Baltimore, 1992. IEEE.

O. Matan, C. J. C. Burges, Y. Le Cun and J. Denker. Multi-digit recognition using a Space Dispacement Neural Network. In J. E. Moody et al., editor, *Advances in Neural Information Processing Systems 4*, Denver, 1992. Morgan Kaufmann.

J. Keeler, D. E. Rumelhart and W-K. Leow. Integrated segmentation and recognition of hand-printed numerals. In R. Lippmann et al., editor, *Advances in Neural Information Processing Systems 3*, pages 557–563, Denver, 1991. Morgan Kaufmann.

Y. Le Cun, L.D. Jackel, B. Boser, J.S. Denker, H.P. Graf, I. Guyon, D. Henderson, R.E. Howard and W. Hubbard. Handwritten digit recognition: Application of neural network chips and automatic learning. *IEEE Communications Magazine*, pages 41–46, November 1989.

A. Weissbuch and Y. Le Cun. Private communication. 1992.

D. Rumelhart et al. Integrated segmentation and recognition of cursive handwriting. In *Third NEC symposium Computational Learning and Cognition*, Princeton, New Jersey, 1992 (to appear).
